# A Prototype for Automatic Recognition of Spontaneous Facial Actions

**M.S. Bartlett, G. Littlewort, B. Braathen, T.J. Sejnowski[1], and J.R. Movellan**
Institute for Neural Computation [1]and Department of Biology
University of California, San Diego
[1]and Howard Hughes Medical Institute at the Salk Institute
Email: marni, gwen, bjorn, terry, javier @inc.ucsd.edu

## Abstract

We present ongoing work on a project for automatic recognition of spontaneous facial actions. Spontaneous facial expressions differ substantially from posed expressions, similar to how continuous, spontaneous speech differs from isolated words produced on command. Previous methods for automatic facial expression recognition assumed images were collected in controlled environments in which the subjects deliberately faced the camera. Since people often nod or turn their heads, automatic recognition of spontaneous facial behavior requires methods for handling out-of-image-plane head rotations. Here we explore an approach based on 3-D warping of images into canonical views. We evaluated the performance of the approach as a front-end for a spontaneous expression recognition system using support vector machines and hidden Markov models. This system employed general purpose learning mechanisms that can be applied to recognition of any facial movement. The system was tested for recognition of a set of facial actions defined by the Facial Action Coding System (FACS). We showed that 3D tracking and warping followed by machine learning techniques directly applied to the warped images, is a viable and promising technology for automatic facial expression recognition. One exciting aspect of the approach presented here is that information about movement dynamics emerged out of filters which were derived from the statistics of images.

## 1 Introduction

Much of the early work on computer vision applied to facial expressions focused on recognizing a few prototypical expressions of emotion produced on command (e.g. "smile"). These examples were collected under controlled imaging conditions with subjects deliberately facing the camera. Extending these systems to spontaneous facial behavior is a critical step forward for applications of this technology. Spontaneous facial expressions differ substantially from posed expressions, similar to how continuous, spontaneous speech differs from isolated words produced on command. Spontaneous facial expressions are mediated by a distinct neural pathway from posed expressions. The pyramidal motor system, originating in the cortical motor strip, drives voluntary facial actions, whereas involuntary, emotional facial expressions appear to originate in a subcortical motor circuit involving

the basal ganglia, limbic system, and the cingulate motor area (e.g. [15]). Psychophysical work has shown that spontaneous facial expressions differ from posed expressions in a number of ways [6]. Subjects often contract different facial muscles when asked to pose an emotion such as fear versus when they are actually experiencing fear. (See Figure 1b.) In addition, the dynamics are different. Spontaneous expressions have a fast and smooth onset, with apex coordination, in which muscle contractions in different parts of the face peak at the same time. In posed expressions, the onset tends to be slow and jerky, and the muscle contractions typically do not peak simultaneously.

Spontaneous facial expressions often contain much information beyond what is conveyed by basic emotion categories, such as happy, sad, or surprised. Faces convey signs of cognitive state such as interest, boredom, and confusion, conversational signals, and blends of two or more emotions. Instead of classifying expressions into a few basic emotion categories, the work presented here attempts to measure the full range of facial behavior by recognizing facial animation units that comprise facial expressions. The system is based on the Facial Action Coding System (FACS) [7].

FACS [7] is the leading method for measuring facial movement in behavioral science. It is a human judgment system that is presently performed without aid from computer vision. In FACS, human coders decompose facial expressions into action units (AUs) that roughly correspond to independent muscle movements in the face (see Figure 1). Ekman and Friesen described 46 independent facial movements, or "facial actions" (Figure 1). These facial actions are analogous to phonemes for facial expression. Over 7000 distinct combinations of such movements have been observed in spontaneous behavior.

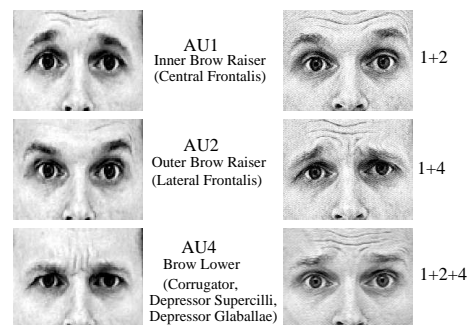

Figure 1: The Facial Action Coding System decomposes facial expressions into component actions. The three individual brow region actions and selected combinations are illustrated. When subjects pose fear they often perform 1+2 (top right), whereas spontaneous fear reliably elicits 1+2+4 (bottom right) [6].

Advantages of FACS include (1) Objectivity. It does not apply interpretive labels to expressions but rather a description of physical changes in the face. This enables studies of new relationships between facial movement and internal state, such as the facial signals of stress or fatigue. (2) Comprehensiveness. FACS codes for all independent motions of the face observed by behavioral psychologists over 20 years of study. (3) Robust link with ground truth. There is over 20 years of behavioral data on the relationships between FACS movement parameters and underlying emotional or cognitive states. Automated facial action coding would be effective for human-computer interaction tools and low bandwidth facial animation coding, and would have a tremendous impact on behavioral science by making objective measurement more accessible.

There has been an emergence of groups that analyze facial expressing into elementary movements. For example, Essa and Pentland [8] and Yacoob and Davis [16] proposed methods to analyze expressions into elementary movements using an animation style coding system inspired by FACS. Eric Petajan's group has also worked for many years on

methods for automatic coding of facial expressions in the style of MPEG4 [5], which codes movement of a set of facial feature points. While coding standards like MPEG4 are useful for animating facial avatars, they are of limited use for behavioral research since, for example, MPEG4 does not encode some behaviorally relevant facial movements such as the muscle that circles the eye (the orbicularis oculi, which differentiates spontaneous from posed smiles [6]). It also does not encode the wrinkles and bulges that are critical for distinguishing some facial muscle activations that are difficult to differentiate using motion alone yet can have different behavioral implications (e.g. see Figure 1b.) One other group has focused on automatic FACS recognition as a tool for behavioral research, lead by Jeff Cohn and Takeo Kanade. They present an alternative approach based on traditional computer vision techniques, including edge detection and optic flow. A comparative analysis of our approaches is available in [1, 4, 10].

## 2 Factorizing rigid head motion from nonrigid facial deformations

The most difficult technical challenge that came with spontaneous behavior was the presence of out-of-plane rotations due to the fact that people often nod or turn their head as they communicate with others. Our approach to expression recognition is based on statistical methods applied directly to filter bank image representations. While in principle such methods may be able to learn the invariances underlying out-of-plane rotations, the amount of data needed to learn such invariances is likely to be impractical. Instead, we addressed this issue by means of deformable 3D face models. We fit 3D face models to the image plane, texture those models using the original image frame, then rotate the model to frontal views, warp it to a canonical face geometry, and then render the model back into the image plane. (See Figures 2,3,4). This allowed us to factor out image variation due to rigid head rotations from variations due to nonrigid face deformations. The rigid transformations were encoded by the rotation and translation parameters of the 3D model. These parameters are retained for analysis of the relation of rigid head dynamics to emotional and cognitive state.

Since our goal was to explore the use of 3D models to handle out-of-plane rotations for expression recognition, we first tested the system using hand-labeling to give the position of 8 facial landmarks.[1] However the approach can be generalized in a straightforward and principled manner to work with automatic 3D trackers, which we are presently developing [9].

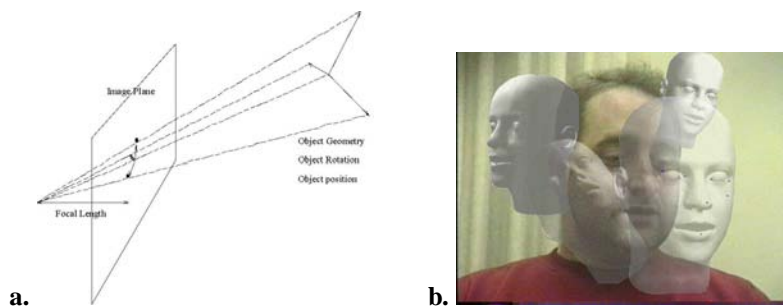

Figure 2: Head pose estimation. a. First camera parameters and face geometry are jointly estimated using an iterative least squares technique b. Next head pose is estimated in each frame using stochastic particle filtering. Each particle is a head model at a particular orientation and scale.

When landmark positions in the image plane are known, the problem of 3D pose estimation is relatively easy to solve. We begin with a canonical wire-mesh face model and adapt it to the face of a particular individual by using 30 image frames in which 8 facial features have been labeled by hand. Using an iterative least squares triangulation technique, we jointly estimate camera parameters and the 3D coordinates of these 8 features. A scattered data interpolation technique is then used to modify the canonical 3D face model so that it fits the 8 feature positions [14]. Once camera parameters and 3D face geometry are known, we use a stochastic particle filtering approach [11] to estimate the most likely rotation and translation parameters of the 3D face model in each video frame. (See [2]).

## 3   Action unit recognition

**Database of spontaneous facial expressions.**   We employed a dataset of spontaneous facial expressions from freely behaving individuals. The dataset consisted of 300 Gigabytes of 640 x 480 color images, 8 bits per pixels, 60 fields per second, 2:1 interlaced. The video sequences contained out of plane head rotation up to 75 degrees. There were 17 subjects: 3 Asian, 3 African American, and 11 Caucasians. Three subjects wore glasses. The facial behaviors in one minute of video per subject were scored frame by frame by 2 teams experts on the FACS system, one lead by Mark Frank at Rutgers, and another lead by Jeffrey Cohn at U. Pittsburgh.

While the database we used was rather large for current digital video storage standards, in practice the number of spontaneous examples of each action unit in the database was relatively small. Hence, we prototyped the system on the three actions which had the most examples: Blinks (AU 45 in the FACS system) for which we used 168 examples provided by 10 subjects, Brow raises (AU 1+2) for which we had 48 total examples provided by 12 subjects, and Brow lower (AU 4) for which we had 14 total examples provided by 12 subjects. Negative examples for each category consisted of randomly selected sequences matched by subject and sequence length. These three facial actions have relevance to applications such as monitoring of alertness, anxiety, and confusion. The system presented here employs general purpose learning mechanisms that can be applied to recognition of any facial action once sufficient training data is available. There is no need to develop special purpose feature measures to recognize additional facial actions.

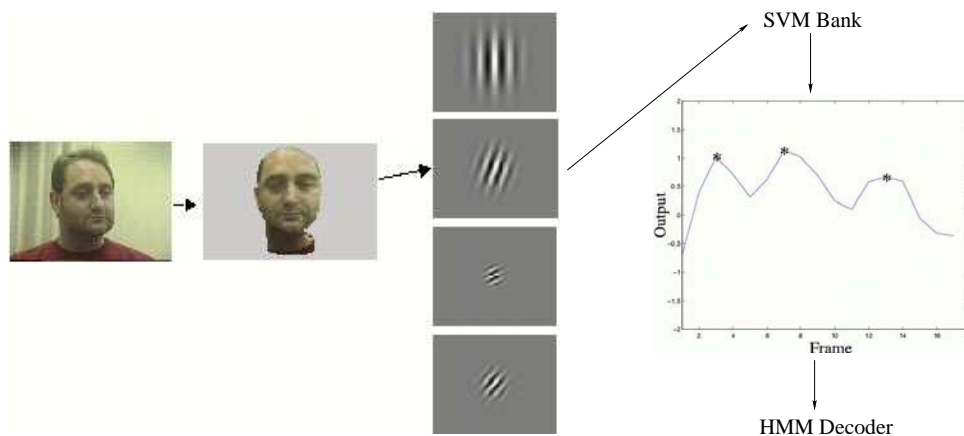

Figure 3: Flow diagram of recognition system. First, head pose is estimated, and images are warped to frontal views and canonical face geometry. The warped images are then passed through a bank of Gabor filters. SVM's are then trained to classify facial actions from the Gabor representation in individual video frames. The output trajectories of the SVM's for full video sequences are then channeled to hidden Markov models.

**Recognition system.** An overview of the recognition system is illustrated in Figure 3. Head pose was estimated in the video sequences using a particle filter with 100 particles. Face images were then warped onto a face model with canonical face geometry, rotated to frontal, and then projected back into the image plane. This alignment was used to define and crop a subregion of the face image containing the eyes and brows. The vertical position of the eyes was 0.67 of the window height. There were 105 pixels between the eyes and 120 pixels from eyes to mouth. Pixel brightnesses were linearly rescaled to [0,255]. Soft histogram equalization was then performed on the image gray-levels by applying a logistic filter with parameters chosen to match the mean and variance of the gray-levels in the neutral frame [13].

The resulting images were then convolved with a bank of Gabor kernels at 5 spatial frequencies and 8 orientations. Output magnitudes were normalized to unit length and then downsampled by a factor of 4. The Gabor representations were then channeled to a bank of support vector machines (SVM's). Nonlinear SVM's were trained to recognize facial actions in individual video frames. The training samples for the SVM's were the action peaks as identified by the FACS experts, and negative examples were randomly selected frames matched by subject. Generalization to novel subjects was tested using leave-one-out cross-validation. The SVM output was the margin (distance along the normal to the class partition). Trajectories of SVM outputs for the full video sequence of test subjects were then channeled to hidden Markov models (HMM's). The HMM's were trained to classify facial actions without using information about which frame contained the action peak. Generalization to novel subjects was again tested using leave-one-out cross-validation.

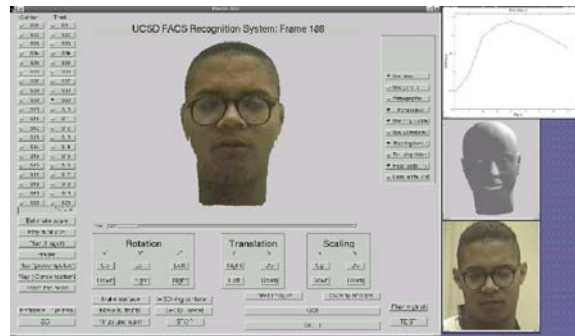

Figure 4: User interface for the FACS recognition system. The face on the bottom right is an original frame from the dataset. Top right: Estimate of head pose. Center image: Warped to frontal view and conical geometry. The curve shows the output of the blink detector for the video sequence. This frame is in the relaxation phase of a blink.

## 4   Results

**Classifying individual frames with SVM's.** SVM's were first trained to discriminate images containing the peak of blink sequences from randomly selected images containing no blinks. A nonlinear SVM applied to the Gabor representations obtained 95.9% correct for discriminating blinks from non-blinks for the peak frames. The nonlinear kernel was of the form $\frac{1}{k+d^2}$ where $d$ is Euclidean distance, and $k$ is a constant. Here $k = 4$.

**Recovering FACS dynamics.** Figure 5a shows the time course of SVM outputs for complete sequences of blinks. Although the SVM was not trained to measure the amount of eye opening, it is an emergent property. In all time courses shown, the SVM outputs are test outputs (the SVM was not trained on the subject shown). Figure 5b shows the SVM trajectory when tested on a sequence with multiple peaks. The SVM outputs provide in-

formation about FACS dynamics that was previously unavailable by human coding due to time constraints. Current coding methods provide only the beginning and end of the action, along with the location and magnitude of the action unit peak. This information about dynamics may be useful for future behavioral studies.

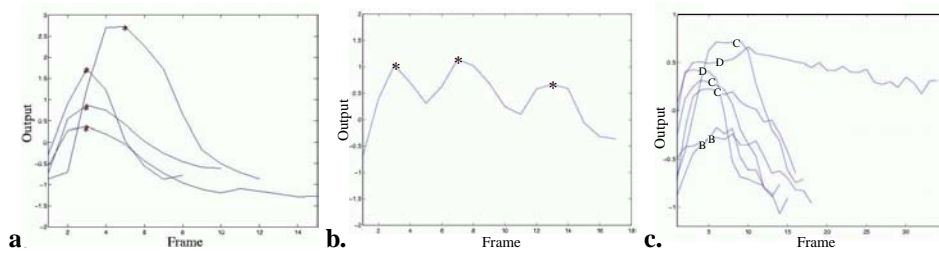

Figure 5: a. Blink trajectories of SVM outputs for four different subjects. Star indicates the location of the AU peak as coded by the human FACS expert. b. SVM output trajectory for a blink with multiple peaks (flutter). c. Brow raise trajectories of SVM outputs for one subject. Letters A-D indicate the intensity of the AU as coded by the human FACS expert, and are placed at the peak frame.

HMM's were trained to classify action units from the trajectories of SVM outputs. HMM's addressed the case in which the frame containing the action unit peak is unknown. Two hidden Markov models, one for Blinks and one for random sequences matched by subject and length, were trained and tested using leave-one-out cross-validation. A mixture of Gaussians model was employed. Test sequences were assigned to the category for which the probability of the sequence given the model was greatest. The number of states was varied from 1-10, and the number of Gaussian mixtures was varied from 1-7. Best performance of 98.2% correct was obtained using 6 states and 7 Gaussians.

**Brow movement discrimination.** The goal was to discriminate three action units localized around the eyebrows. Since this is a 3-category task and SVMs are originally designed for binary classification tasks, we trained a different SVM on each possible binary decision task: Brow Raise (AU 1+2) versus matched random sequences, Brow Lower (AU 4) versus another set of matched random sequences, and Brow Raise versus Brow Lower. The output of these three SVM's was then fed to an HMM for classification. The input to the HMM consisted of three values which were the outputs of each of the three 2-category SVM's. As for the blinks, the HMM's were trained on the "test" outputs of the SVM's. The HMM's achieved 78.2% accuracy using 10 states, 7 Gaussians and including the first derivatives of the observation sequence in the input. Separate HMM's were also trained to perform each of the 2-category brow movement discriminations in image sequences. These results are summarized in Table 1.

Figure 5c shows example output trajectories for the SVM trained to discriminate Brow Raise from Random matched sequences. As with the blinks, we see that despite not being trained to indicate AU intensity, an emergent property of the SVM output was the magnitude of the brow raise. Maximum SVM output for each sequence was positively correlated with action unit intensity, as scored by the human FACS expert ($r = .43, t(42) = 3.1, p = 0.0017$).

The contribution of Gabors was examined by comparing linear and nonlinear SVM's applied directly to the difference images versus to Gabor outputs. Consistent with our previous findings [12], Gabor filters made the space more linearly separable than the raw difference images. For blink detection, a linear SVM on the Gabors performed significantly better (93.5%) than a linear SVM applied directly to difference images (78.3%). Using a nonlinear SVM with difference images improved performance substantially to 95.9%, whereas the nonlinear SVM on Gabors gave only a small increment in performance, also

| Action | % Correct (HMM) | N |
|---|---|---|
| Blink vs. Non-blink | 98.2 | 168 |
| Brow Raise vs. Random | 90.6 | 48 |
| Brow Lower vs. Random | 75.0 | 14 |
| Brow Raise vs. Brow Lower | 93.5 | 31 |
| Brow Raise vs. Lower vs. Random | 78.2 | 62 |

Table 1: Summary of results. All performances are for generalization to novel subjects. Random: Random sequences matched by subject and length. N: Total number of positive (and also negative) examples.

to 95.9%. A similar pattern was obtained for the brow movements, except that nonlinear SVMs applied directly to difference images did not perform as well as nonlinear SVM's applied to Gabors. The details of this analysis, and also an analysis of the contribution of SVM's to system performance, are available in [1].

# 5   Conclusions

We explored an approach for handling out-of-plane head rotations in automatic recognition of spontaneous facial expressions from freely behaving individuals. The approach fits a 3D model of the face and rotates it back to a canonical pose (e.g., frontal view). We found that machine learning techniques applied directly to the warped images is a promising approach for automatic coding of spontaneous facial expressions.

This approach employed general purpose learning mechanisms that can be applied to the recognition of any facial action. The approach is parsimonious and does not require defining a different set of feature parameters or image operations for each facial action. While the database we used was rather large for current digital video storage standards, in practice the number of spontaneous examples of each action unit in the database was relatively small. We therefore prototyped the system on the three actions which had the most examples. Inspection of the performance of our system shows that 14 examples was sufficient to successfully learn an action, an order of 50 examples was sufficient to achieve performance over 90%, and an order of 150 examples was sufficient to achieve over 98% accuracy and learn smooth trajectories. Based on these results, we estimate that a database of 250 minutes of coded, spontaneous behavior would be sufficient to train the system on the vast majority of facial actions.

One exciting finding is the observation that important measurements emerged out of filters derived from the statistics of the images. For example, the output of the SVM filter matched to the blink detector could be potentially used to measure the dynamics of eyelid closure, even though the system was not designed to explicitly detect the contours of the eyelid and measure the closure. (See Figure 5.)

The results presented here employed hand-labeled feature points for the head pose tracking step. We are presently developing a fully automated head pose tracker that integrates particle filtering with a system developed by Matthew Brand for automatic real-time 3D tracking based on optic flow [3].

All of the pieces of the puzzle are ready for the development of automated systems that recognize spontaneous facial actions at the level of detail required by FACS. Collection of a much larger, realistic database to be shared by the research community is a critical next step.

# Acknowledgments

Support for this project was provided by ONR N00014-02-1-0616, NSF-ITR IIS-0220141 and IIS-0086107, DCI contract No.2000-I-058500-000, and California Digital Media Innovation Program DiMI 01-10130.

## Footnotes

[1] Although human labeling can be highly precise, the labels employed here had substantial error due to inattention when the face moved. Mean deviation between two labelers was 4 pixels $\pm 8.7$. Hence it may be realistic to suppose that a fully automatic head pose tracker may achieve at least this level of accuracy.

# References

[1] M.S. Bartlett, B. Braathen, G. Littlewort-Ford, J. Hershey, I. Fasel, T. Marks, E. Smith, T.J. Sejnowski, and J.R. Movellan. Automatic analysis of of spontaneous facial behavior: A final project report. Technical Report UCSD MPLab TR 2001.08, University of California, San Diego, 2001.

[2] B. Braathen, M.S. Bartlett, G. Littlewort-Ford, and J.R. Movellan. 3-D head pose estimation from video by nonlinear stochastic particle filtering. In *Proceedings of the 8th Joint Symposium on Neural Computation*, 2001.

[3] M. Brand. Flexible flow for 3d nonrigid tracking and shape recovery. In *CVPR*, 2001.

[4] J.F. Cohn, T. Kanade, T. Moriyama, Z. Ambadar, J. Xiao, J. Gao, and H. Imamura. A comparative study of alternative FACS coding algorithms. Technical Report CMU-RI-TR-02-06, Robotics Institute, Carnegie-Mellon Univerisity, 2001.

[5] P. Doenges, F. Lavagetto, J. Ostermann, I.S. Pandzic, and E. Petajan. Mpeg-4: Audio/video and synthetic graphics/audio for real-time, interactive media delivery. *Image Communications Journal*, 5(4), 1997.

[6] P. Ekman. *Telling Lies: Clues to Deceit in the Marketplace, Politics, and Marriage*. W.W. Norton, New York, 3rd edition, 2001.

[7] P. Ekman and W. Friesen. *Facial Action Coding System: A Technique for the Measurement of Facial Movement*. Consulting Psychologists Press, Palo Alto, CA, 1978.

[8] I. Essa and A. Pentland. Coding, analysis, interpretation, and recognition of facial expressions. *IEEE Transactions on Pattern Analysis and Machine Intelligence*, 19(7):757–63, 1997.

[9] I.R. Fasel, M.S. Bartlett, and J.R. Movellan. A comparison of gabor filter methods for automatic detection of facial landmarks. In *Proceedings of the 5th International Conference on Face and Gesture Recognition*, 2002. Accepted.

[10] M.G. Frank, P. Perona, and Y. Yacoob. Automatic extraction of facial action codes. final report and panel recommendations for automatic facial action coding. Unpublished manuscript, Rutgers University, 2001.

[11] G. Kitagawa. Monte carlo filter and smoother for non-Gaussian nonlinear state space models. *Journal of Computational and Graphical Statistics*, 5(1):1–25, 1996.

[12] G. Littlewort-Ford, M.S. Bartlett, and J.R. Movellan. Are your eyes smiling? detecting genuine smiles with support vector machines and gabor wavelets. In *Proceedings of the 8th Joint Symposium on Neural Computation*, 2001.

[13] J.R. Movellan. Visual speech recognition with stochastic networks. In G. Tesauro, D.S. Touretzky, and T. Leen, editors, *Advances in Neural Information Processing Systems*, volume 7, pages 851–858. MIT Press, Cambridge, MA, 1995.

[14] Frédéric Pighin, Jamie Hecker, Dani Lischinski, Richard Szeliski, and David H. Salesin. Synthesizing realistic facial expressions from photographs. *Computer Graphics*, 32(Annual Conference Series):75–84, 1998.

[15] W. E. Rinn. The neuropsychology of facial expression: A review of the neurological and psychological mechanisms for producing facial expressions. *Psychological Bulletin*, 95(1):52–77, 1984.

[16] Y. Yacoob and L. Davis. Recognizing human facial expressions from long image sequences using optical flow. *IEEE Transactions on Pattern Analysis and Machine Intelligence*, 18(6):636–642, 1996.
